# Discovering hidden features with Gaussian processes regression

**Francesco Vivarelli**
Centro Ricerche Ambientali
Montecatini,
via Ciro Menotti, 48
48023 Marina di Ravenna
Italy
`fvivarelli@cramont.it`

**Christopher K. I. Williams**
Division of Informatics
The University of Edinburgh,
5 Forrest Hill,
Edinburgh, EH1 2QL
United Kingdom
`ckiw@dai.ed.ac.uk`

## Abstract

In Gaussian process regression the covariance between the outputs at input locations $\mathbf{x}$ and $\mathbf{x}'$ is usually assumed to depend on the distance $(\mathbf{x} - \mathbf{x}')^T W (\mathbf{x} - \mathbf{x}')$, where $W$ is a positive definite matrix. $W$ is often taken to be diagonal, but if we allow $W$ to be a general positive definite matrix which can be tuned on the basis of training data, then an eigen-analysis of W shows that we are effectively creating hidden features, where the dimensionality of the hidden-feature space is determined by the data. We demonstrate the superiority of predictions using the general matrix over those based on a diagonal matrix on two test problems.

## 1 Introduction

Over the last few years Bayesian approaches to prediction with neural networks have come to the fore. Following an argument in Neal (1996) concerning the equivalence between infinite neural networks and certain Gaussian processes, Gaussian process (GP) prediction has also become popular, and Rasmussen (1996) has demonstrated good performance of GP predictors on a number of tasks.

In Gaussian process prediction as applied by Rasmussen (1996), Williams and Rasmussen (1996) and others, the covariance between the outputs at locations $\mathbf{x}$ and $\mathbf{x}'$ is usually assumed to depend on the distance $(\mathbf{x} - \mathbf{x}')^T W (\mathbf{x} - \mathbf{x}')$, where $W$ is a positive definite, *diagonal* matrix. This means that different dimensions in the input space can have different relevances to the prediction problem (c.f.MacKay and Neal's idea of Automatic Relevance Determination (Neal, 1996)). However, some of the reasoning about the success of neural networks and methods such as projection pursuit regression suggests that discovering relevant *directions* in feature space is important; clearly the ARD model is a special case, where these directions

are parallel to the axes in the input feature space. In this paper we allow $W$ to be a general positive semidefinite matrix (defining a Mahalanobis distance in the input space), thereby allowing general directions in the input space to be selected. We then compare the performance of GP predictors using the diagonal and full distance matrices on some regression problems.

The structure of the paper is as follows. GPs for regression are introduced in Section 2, where we also explain the rôle played by the distance matrix $W$ and the criterion used to compare the generalisation performances of the diagonal and the general distance matrices. The two methods have been compared on two regression tasks and the results of our experiments are shown in Section 3. A summary of the work done and some open questions are presented in Section 4.

## 2 Gaussian processes and prediction

In this paper we use Gaussian process models as predictors. Consider a stochastic process $Y(\mathbf{x})$, with the input observable $\mathbf{x}$ belonging to some input space $\mathcal{X} \subseteq \mathbb{R}^d$. Gaussian processes are a subset of stochastic processes that can be defined by specifying the mean and covariance functions, $\mu(\mathbf{x}) = \mathcal{E}[Y(\mathbf{x})]$ and $C_p(\mathbf{x}, \mathbf{x}') = \mathcal{E}[Y(\mathbf{x}) Y(\mathbf{x}')]$ respectively. For the work below we shall set $\mu(\mathbf{x}) \equiv 0$. Although the GP formulation provides a prior over functions, for our purposes it suffices to note that the $y$-values $Y(\mathbf{x}^1), Y(\mathbf{x}^2), \ldots, Y(\mathbf{x}^n)$ corresponding to $\mathbf{x}$-values $\mathbf{x}^1, \mathbf{x}^2, \ldots, \mathbf{x}^n$ have a multivariate Gaussian distribution $\mathcal{N}(\mathbf{0}, K_p)$, where $(K_p)_{ij} = C_p(\mathbf{x}^i, \mathbf{x}^j)$. The specific form of the covariance function that we shall use is

$$C_p(\mathbf{x}, \mathbf{x}') = \sigma_p^2 \exp\left[-\frac{1}{2}(\mathbf{x} - \mathbf{x}')^T W(\mathbf{x} - \mathbf{x}')\right]. \tag{1}$$

When $W$ is a diagonal matrix the entry $w_{ii}$ is the inverse of the squared *correlation length-scale* of the process along the direction $i$. In particular, we note that this model is closely related to the Automatic Relevance Determination method of MacKay and Neal (Neal, 1996), as a small lengthscale along a certain direction of the space highlights the relevance of the corresponding input feature (assuming that the inputs are normalised).

For the prediction problem, let us suppose to have $n$ data points $\mathcal{D}_n = \left\{\left(\mathbf{x}^1, t^1\right), \left(\mathbf{x}^2, t^2\right), \ldots, \left(\mathbf{x}^n, t^n\right)\right\}$, where $t^i$ is the output-value corresponding to the input $\mathbf{x}^i$. The $t$'s are assumed to be generated from the true $y$-values by adding Gaussian noise of variance $\sigma_\nu^2$. Given the assumption of a Gaussian process prior over functions, it is a standard result (e.g. Whittle, 1963) that the predictive distribution $p(t|\mathbf{x}, \mathcal{D}_n)$ corresponding to a new input is $\mathcal{N}(\hat{y}(\mathbf{x}), \sigma^2(\mathbf{x}))$, with mean and variance

$$\hat{y}(\mathbf{x}) = \mathbf{k}^T(\mathbf{x}) K^{-1}\mathbf{t} \tag{2}$$

$$\sigma^2(\mathbf{x}) = C_p(\mathbf{x}, \mathbf{x}) + \sigma_\nu^2 - \mathbf{k}^T(\mathbf{x}) K^{-1}\mathbf{k}(\mathbf{x}), \tag{3}$$

where $K = K_p + \sigma_\nu^2 I$, $\mathbf{k}^T(\mathbf{x}) = \left(C_p(\mathbf{x}, \mathbf{x}^1), C_p(\mathbf{x}, \mathbf{x}^2), \ldots, C_p(\mathbf{x}, \mathbf{x}^n)\right)$ and $\mathbf{t}^T = \left(t^1, t^2, \ldots t^n\right)$.

This method of prediction assumes that the process $y(\mathbf{x})$ we are modelling is really a function of the observable $\mathbf{x}$. However it is often the case that for real world problems the $y$ is actually a function of a set of hidden features $\mathbf{z} \in \mathcal{Z} \subseteq \mathbb{R}^q$ which arise from a combination of the manifest variables $\mathbf{x}$. In particular we wish to study the problem in which the hidden features are a linear combination of the observable coordinates through a $q \times d$ matrix $M$, where $q < d$ (i.e. $\mathbf{z} = M\mathbf{x}$). In this case,

the covariance of the function $y$ is specified by Equation 1 but turns out to depend upon the estimation of the distance between hidden features $(\mathbf{z} - \mathbf{z}')^T \Psi (\mathbf{z} - \mathbf{z}')$. Since $\mathbf{z} = M\mathbf{x}$, $(\mathbf{z} - \mathbf{z}') = M(\mathbf{x} - \mathbf{x}')$ and $W = M^T \Psi M$.

A GP model depends on the parameters which describe the covariance function (i.e. $\sigma_p^2$, $\sigma_\nu^2$ and the elements of $W$). The training of a GP can be carried out by either estimating the parameters of the covariance function (for example, using the maximum likelihood method) or using a Bayesian approach and sampling from the posterior distribution over the parameters (Williams and Rasmussen, 1996). We follow the first approach, maximising the logarithm of the likelihood

$$\mathcal{L} = \log p(\mathcal{D}_n | \theta) = -\frac{1}{2} \log \det K - \frac{1}{2} \mathbf{t}^T K^{-1} \mathbf{t} - \frac{n}{2} \log 2\pi \qquad (4)$$

where $K^{-1}$ depends upon $\theta$, the vector of parameters of the covariance function.

The number of free parameters depends on the number of non-zero elements of the matrix $W$. Usually, $W$ is chosen to be diagonal and the number of free parameters is $d + 2$ (the $d$ diagonal elements, $\sigma_p^2$ and $\sigma_\nu^2$). We notice that this parametrisation of $W$ allows the discovery of relevant directions in the observed space; it does not lead to an estimation of a general mapping of $\mathcal{X}$ onto the feature space $\mathcal{Z}$ as the relevant directions are parallel to the axes in the input manifest space.

If $q$ is not known in advance, it is preferable to use a general symmetric positive semidefinite matrix $W$. A parametrisation of such a matrix follows from the Choleski decomposition as $W = U^T U$, where $U$ is an upper triangular matrix with positive entries on the diagonal (Williams, 1996). Hence the factorisation of $U$ turns out to be

$$U = \begin{pmatrix} \exp[u_{1,1}] & u_{1,2} & \dots & u_{1,d} \\ 0 & \exp[u_{2,2}] & \dots & u_{2,d} \\ 0 & 0 & \dots & u_{3,d} \\ \dots & \dots & \dots & \exp[u_{d,d}] \end{pmatrix}. \qquad (5)$$

The elements on the diagonal are positive because of the exponential. Being symmetric, $W$ has at most $d(d+1)/2$ independent entries and thus the total number of free parameters of the GP model is $2 + d(d+1)/2$.

We note that such a full distance matrix $W$ allows an estimation of the matrix $M$ from an eigenvalue decomposition of $W = V\Lambda V^T$, where $\Lambda$ is a diagonal matrix of the eigenvalues of $W$ and $V$ is the matrix of the eigenvectors. The dimension of the hidden feature space $\mathcal{Z}$ can be inferred by the number of relevant eigenvalues of the matrix $\Lambda$ (which are the inverse of the squared correlation lengths of the process along the directions of the hidden space). The directions of the hidden feature space are defined by the eigenvectors corresponding to the relevant eigenvalues; in particular the matrix composed by these eigenvectors gives an estimate of the mapping from $\mathcal{X}$ to $\mathcal{Z}$. In the following the diagonal and the general full correlation matrices are designated by $W_d$ and $W_f$.

It is important to observe that the predictor obtained using $W_f$ is not equivalent to an additive model (Hastie and Tibshirani, 1990), as the predictor is a multivariate function of $\mathbf{z}$ rather than being an additive function of the components of $\mathbf{z}$. However, it would be possible to produce an additive function in the GP context, using a covariance function which is the sum of one-dimensional covariance functions based on projections of $\mathbf{x}$.

## 2.1 Generalisation error

Consider predicting the value of a function $y(\mathbf{x})$ with a predictor $\hat{y}(\mathbf{x})$. A commonly-used measure of the generalisation error given a dataset $\mathcal{D}_n$ is the average squared

error

$$E^g\left(\mathcal{D}_n\right) = \int \left(y\left(\mathbf{x}\right) - \hat{y}_{\mathcal{D}_n}\left(\mathbf{x}\right)\right)^2 p\left(\mathbf{x}\right) d\mathbf{x}. \tag{6}$$

The average generalisation error $E^g\left(n\right)$ for a dataset of size $n$ is obtained by averaging over the choice of training dataset, i.e. $E^g\left(n\right) = \mathcal{E}_\mathcal{D}\left[E^g\left(\mathcal{D}_n\right)\right]$. $E^g\left(\mathcal{D}_n\right)$ can sometimes be evaluated analytically or by numerical integration, but it is usually necessary to use samples to perform the average over training datasets $\mathcal{D}_n$.

In order to investigate the generalisation capabilities of GPs using a diagonal and full distance matrices $W_d$ and $W_f$, we trained the GP predictors on some regression tasks. The generalisation errors are compared by looking at the relative error

$$\rho\left(\mathcal{D}_n\right) = \frac{E^g_d\left(\mathcal{D}_n\right) - E^g_f\left(\mathcal{D}_n\right)}{E^g_d\left(\mathcal{D}_n\right)} \tag{7}$$

where $E^g_d\left(\mathcal{D}_n\right)$ and $E^g_f\left(\mathcal{D}_n\right)$ are the generalisation errors reported using a diagonal and a full distance matrix respectively. This ratio allow us to perform a fair comparison between the pairwise differences of the generalisation errors for each dataset and the actual value $E^g_d\left(\mathcal{D}_n\right)$. The expected value $\rho\left(n\right)$ is the average over the sampling of the training data $\mathcal{D}_n$: $\rho\left(n\right) = \mathcal{E}_\mathcal{D}\left[\rho\left(\mathcal{D}_n\right)\right]$.

## 3 Results

We have conducted experiments to compare the generalisation capabilities of a GP predictor with full and diagonal distance matrices. In this section we illustrate the results we obtained by training a GP on two regression tasks, the regression of a trigonometric function (Section 3.1), and the regression of a high-interaction surface (Section 3.2).

### 3.1 Regression of a trigonometric function

In the first experiments, a GP has been trained on observations drawn from the function $y\left(z\right) = \sin\left(2\pi z\right)$ corrupted by Gaussian noise of mean zero and variance $\sigma_\nu^2 = 10^{-4}, 10^{-3}, 10^{-2}, 10^{-1}, 1$. The hidden feature $z \in \mathbb{R}$ has been generated from the observable variables $\mathbf{x} \in \mathbb{R}^2$ through the transformation $z = \mathbf{m}^T\mathbf{x}$, where $\mathbf{m}^T = \left(1/\sqrt{2}, 1/\sqrt{2}\right)$ and $\mathbf{x} \sim \mathcal{N}\left(\mathbf{0}, \mathbb{I}\right)$. We wish to infer the process $y\left(z\right)$ (which is actually a function of the one-dimensional feature $z$) by using a GP on the manifest space $\mathbb{R}^2$. We evaluated the expected generalisation errors of Equation 6 by Gaussian quadrature (Press et al., 1992) and estimated the expected relative error $\rho\left(n\right)$ by averaging over 10 different samples of the training set.

The parameters of the covariance function are optimised on each of the 10 training datasets by maximising the likelihood (see Equation 4) with the conjugate gradient algorithm (Press et al., 1992) with 50 (for $W_d$) and 70 (for $W_f$) iterations for the largest training sets with 256 data.

Figure 1 reports the value of $\rho\left(n\right)$ on the vertical axis as a function of the amount of training data ($x$ axis). The variance of the noise has been set to 0.01 in Figure 1(a) and 0.1 in Figure 1(b).

The plots show that the use of $W_f$ significantly improves the generalisation performance with respect to a diagonal matrix as the relative error $\rho\left(n\right)$ lies well above zero, within its confidence interval. This is particularly highlighted in Figure 1(a) where for datasets larger than 32 data, $\rho\left(n\right)$ is larger than 75%. We notice that for small datasets, $\rho\left(n\right)$ is close to zero, as the distribution of its values are spread

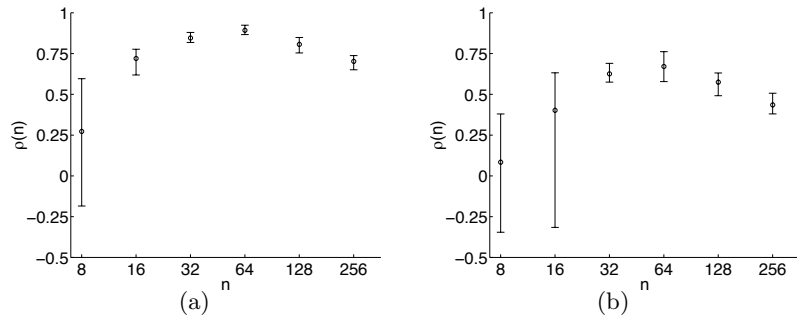

Figure 1: The Figures report on the y axis the graphs of $\rho(n)$ (see Equation 7) as a function of the amount of training data (x axis); the noise level is set to 0.01 (Figure 1(a)) and to 0.1 (Figure 1(b)). The error bars are generated by the minimum and the maximum value of $\rho(\mathcal{D}_n)$ which occurred over the 10 training datasets.

out around zero with wide confidence intervals. This is due to the fact that with small amounts of data it is not possible to train the GP properly; in particular, as the number of free parameters of $W_f$ is larger than that of $W_d$, the former needs larger datasets for the training than the latter in order to avoid overfitting. A fully Bayesian treatment of the training of a GP (see Section 2) would not be so seriously affected by this problem since the prediction of the GP would be marginalised over the posterior distribution of the parameters. For large datasets, the relative error declines after having reached its maximum value; this agrees with the intuition that with large amounts of data, both methods will be become good predictors. Similar remarks apply also to Figure 1(b) (where $\sigma_\nu^2 = 0.1$) although we notice that the relative error $\rho(n)$ assumes lower values due to the higher noise variance.

The better perfomance of $W_f$ with respect to $W_d$ can be explained by an eigenanalysis of the two distance matrices. Since one eigenvalue of $W_f$ is much larger than the other ($O(10)$ vs. $O(10^{-4})$), the full rank distance matrix is able to discover the relevant true dimension of the process. The eigenvector corresponding to the larger eigenvalue represents the operator which maps the space of the observables onto the hidden feature space. $W_d$ fails to find out the effective dimension of the problem as it is characterised by two eigenvalues of similar magnitude ($O(10)$).

## 3.2 A high-interaction surface

We also tested our method on an example taken from Breiman (1993) which is concerned with a regression problem of a surface in a high dimensional space. The target function is $y(\mathbf{x}) = \sigma(z_1) + \sigma(z_2) + \sigma(z_3)$, where $\sigma(z)$ is the sigmoid function $\sigma(z) = \exp[z] / (1 + \exp[z])$. The hidden features $z_1$, $z_2$ and $z_3$ are derived from the transformation $z_i = 2(l_i - 2)$, $i = 1 \ldots 3$, where the $l_i$ are the normalised inner products $\mathbf{m}_i^T \mathbf{x}$. The observed variables $\mathbf{x} \in \mathbb{R}^{10}$ are uniformly distributed over $[0,1]^{10}$; the three vectors $\mathbf{m}_i$ are $\mathbf{m}_1^T = (10, 9, 3, 7, -6, -5, -9, -3, -2, -1)$, $\mathbf{m}_2^T = (-1, -2, -3, -4, -5, -6, 7, 8, 9, 10)$ and $\mathbf{m}_3^T = (-1, -2, -3, 4, 5, 4, -3, -2, -1, 0)$. The values of the true function are also corrupted by Gaussian noise of mean zero; the variance of the noise was such that the ratio between the standard deviations of the signal $y(\mathbf{x})$ and the the noise was 4.0, as in Breiman (1993).

We have run experiments, training GPs with diagonal and full distance matrices on 10 data sets of size 64, 128, 256 and 512; in his work, Breiman used training

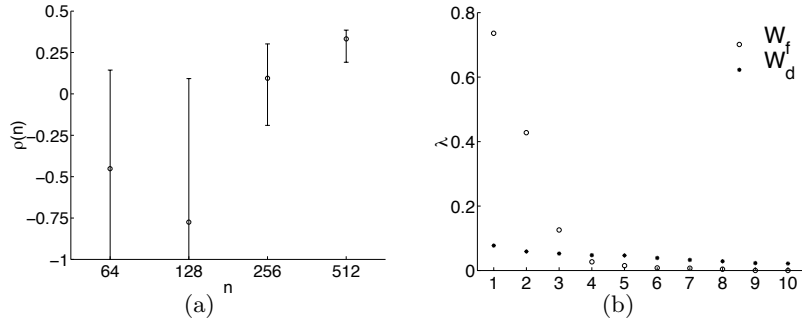

Figure 2: Figure 2(a) reports on the y axis the graph of $\rho\left(n\right)$ (see Equation 7) as a function of the amount of training data (x axis); the error bars are generated by the minimum and the maximum value of $\rho\left(\mathcal{D}_n\right)$ that occurred over the 10 training datasets. Figure 2(a) shows the graph of the ten eigenvalues of the $W_d$ ($*$) and the $W_f$ ($\circ$) distance matrices obtained using one training set of 512 data. The lower values reached by training sets with 64 and 128 data are $-1.06$ and $-1.97$ respectively.

sets with 400 datapoints. The GP's parameters are optimised on each of the 10 training datasets by maximising the likelihood (see Equation 4) with the conjugate gradient algorithm (Press et al., 1992). The generalisation errors of $W_d$ and $W_f$ have been estimated using 1024 test data points; the relative generalisation error $\rho\left(n\right)$ (c.f.Equation 7) is shown in Figure 2(a). We observe that for datasets of size 512 the use of $W_f$ significantly reduces the relative error with respect the diagonal matrix. Models trained with smaller training sets do not have such good generalisation performance because the larger number of parameters in $W_f$ (57) overfits the data.

An eigenvalue decomposition of the distance matrices shows that $W_f$ is able to discover the underlying structure of the process. Figure 2(b) displays the eigenvalues of $W_f$ and $W_d$ optimised for one of the training sets of 512 data. $W_f$ is characterised by three large eigenvalues, whose eigenvectors indicate the three main directions in the feature space; thus the full matrix is able to find out three out of ten directions which are responsible of the variation of the function. Conversely, $W_d$ fails to discover the hidden features in the data; since all the eigenvalues have almost the same magnitude, all the input dimensions of the observed variable are equally relevant in training the GP.

The eigenvectors $\mathbf{e}_i^f, i = 1, 2, 3$ of $W_f$ define a basis in the space generating a subspace of features. In order to verify the subspace spanned by the $\mathbf{e}_i^f$ actually overlaps the hidden feature space, we tried to express the former set of vectors as a linear combination the latter. Thus we computed the singular values (Press et al., 1992) of the matrix composed by the normalised vectors $\mathbf{m}_i$ and the basis $\mathbf{e}_i^f$'s. As three out of six singular values are negligible with respect to the others ($O\left(10^{-2}\right)$ vs. $O\left(1\right)$), the original hidden transformation can be well approximated as a linear combination of the new basis of eigenvectors showing that the eigenspace of $W_f$ is a good approximation of the hidden feature space.

# 4 Discussion

In this paper we have shown how to discover hidden features with GP regression. We also note that this technique could be applied to problems where Gaussian process predictors are used in classification problems. An attractive feature of the method is that it allows the appropriate dimensionality of the $\mathbf{z}$ space to be discovered. If we wish to restrict the maximum dimensionality of $\mathcal{Z}$ to be $q$ then one could use a distance matrix of rank-$q$, i.e.$(\Psi^{\frac{1}{2}}M)^T(\Psi^{\frac{1}{2}}M)$.

The idea of allowing a general transformation of the input space has been mentioned before in the literature, for example in (Girosi et al., 1995). However, Girosi *et al* suggest setting the parameters in $W_f$ by cross-validation; we believe that this is not very practical in high-dimensional spaces. The results obtained show that the use of a full distance matrix can reduce significantly the relative error with respect to the use of a diagonal distance matrix. As the training of the GP has been carried out maximising the logarithm of the likelihood, this effect was particularly evident when larger amounts of data were used; this problem can be reduced when a full Bayesian approach to the GP regression is used.

Currently we are investigating how the input-dimensionality of the affects GP regression with a general distance matrix $W$ (for a fixed dimensionality of $\mathcal{Z}$).

### Acknowledgements

This research forms part of the "Validation and Verification of Neural Network Systems" project funded jointly by EPSRC (GR/K 51792) and British Aerospace. We thank Dr. Andy Wright of BAe for helpful discussions.

# References

Breiman, L. (1993). Hinging hyperplanes for regression, classification and function approximation. *IEEE Trans. on Information Theory*, 39(3):999–1013.

Girosi, F., Jones, M., and Poggio, T. (1995). Regularization Theory and Neural Networks Architectures. *Neural Computation*, 7(2):219–269.

Hastie, T. J. and Tibshirani, R. J. (1990). *Generalized Additive Models*. Chapman and Hall, London.

Neal, R. M. (1996). *Bayesian Learning for Neural Networks*. Springer. Lecture Notes in Statistics 118.

Press, W., Teukolsky, S., Vetterling, W., and Flannery, B. (1992). *Numerical Recipes in C. The Art of Scientific Computing*. Cambridge University Press. second edition.

Rasmussen, C. E. (1996). *Evaluation of Gaussian Processes and Other Methods for Nonlinear Regression*. PhD thesis, Dept. of Computer Science, University of Toronto.

Whittle, P. (1963). *Prediction and regulation by linear least square methods*. English Universities Press.

Williams, C. K. I. and Rasmussen, C. E. (1996). Gaussian processes for regression. In Touretzky, M. C. and Mozer, M. C. and Hasselmo, M. E., editors, *Advances in Neural Information Processing Systems 8*, pages 514–520. MIT Press.

Williams, P. M. (1996). Conditional multivariate densities. *Neural Computation*, 8(4).
